# Adaptive Manifold Learning

**Jing Wang,   Zhenyue Zhang**
Department of Mathematics
Zhejiang University, Yuquan Campus,
Hangzhou, 310027, P. R. China
wroaring@sohu.com
zyzhang@zju.edu.cn

**Hongyuan Zha**
Department of Computer Science
Pennsylvania State University
University Park, PA 16802
zha@cse.psu.edu

## Abstract

Recently, there have been several advances in the machine learning and pattern recognition communities for developing manifold learning algorithms to construct nonlinear low-dimensional manifolds from sample data points embedded in high-dimensional spaces. In this paper, we develop algorithms that address two key issues in manifold learning: 1) the adaptive selection of the neighborhood sizes; and 2) better fitting the local geometric structure to account for the variations in the curvature of the manifold and its interplay with the sampling density of the data set. We also illustrate the effectiveness of our methods on some synthetic data sets.

## 1   Introduction

Recently, there have been advances in the machine learning community for developing effective and efficient algorithms for constructing nonlinear low-dimensional manifolds from sample data points embedded in high-dimensional spaces, emphasizing simple algorithmic implementation and avoiding optimization problems prone to local minima. The proposed algorithms include Isomap [6], locally linear embedding (LLE) [3] and its variations, manifold charting [1], hessian LLE [2] and local tangent space alignment (LTSA) [7], and they have been successfully applied in several computer vision and pattern recognition problems. Several drawbacks and possible extensions of the algorithms have been pointed out in [4, 7] and the focus of this paper is to address two key issues in manifold learning: 1) how to adaptively select the neighborhood sizes in the k-nearest neighbor computation to construct the local connectivity; and 2) how to account for the variations in the curvature of the manifold and its interplay with the sampling density of the data set. We will discuss those two issues in the context of local tangent space alignment (LTSA) [7], a variation of locally linear embedding (LLE) [3] (see also [5],[1]). We believe the basic ideas we proposed can be similarly applied to other manifold learning algorithms.

We first outline the basic steps of LTSA and illustrate its failure modes using two simple examples. Given a data set $X = [x_1, \ldots, x_N]$ with $x_i \in \mathcal{R}^m$, sampled (possibly with noise) from a $d$-dimensional manifold ($d < m$), LTSA proceeds in the following steps.

1) LOCAL NEIGHBORHOOD CONSTRUCTION. For each $x_i$, $i = 1, \ldots, N$, determine a set $X_i = [x_{i_1}, \ldots, x_{i_{k_i}}]$ of its neighbors ($k_i$ nearest neighbors, for example).

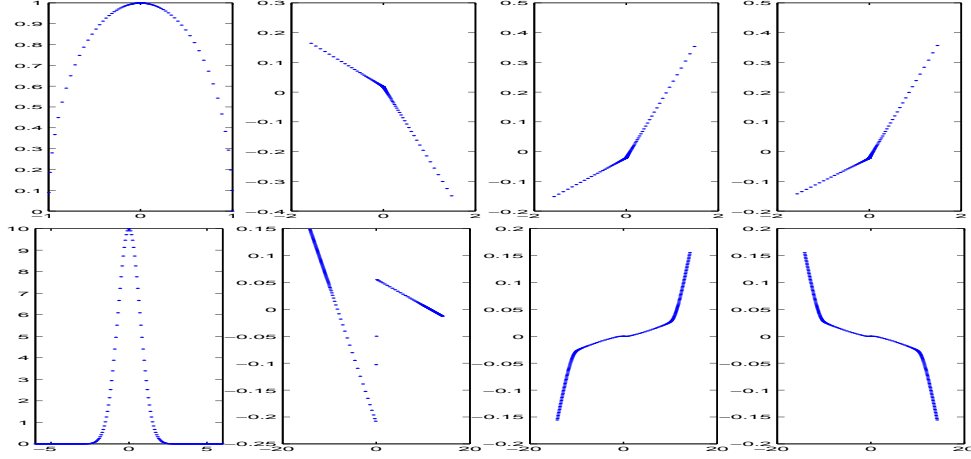

Figure 1: The data sets (first column) and computed coordinates $\tau_i$ by LTSA vs. the centered arc-length coordinates Top row: Example 1. Bottom row: Example 2.

2) LOCAL LINEAR FITTING. Compute an orthonormal basis $Q_i$ for the $d$-dimensional tangent space of the manifold at $x_i$, and the orthogonal projection of each $x_{i_j}$ to the tangent space: $\theta_j^{(i)} = Q_i^T(x_{i_j} - \bar{x}_i)$ where $\bar{x}_i$ is the mean of the neighbors.

3) LOCAL COORDINATES ALIGNMENT. Align the $N$ *local* projections $\Theta_i = [\theta_1^{(i)}, \cdots, \theta_{k_i}^{(i)}]$, $i = 1, \ldots, N$, to obtain the global coordinates $\tau_1, \ldots, \tau_N$. Such an alignment is achieved by minimizing the global reconstruction error

$$\sum_i \|E_i\|_2^2 \equiv \sum_i \|T_i(I - \frac{1}{k_i}ee^T) - L_i\Theta_i\|_2^2 \tag{1.1}$$

over all possible $L_i \in \mathcal{R}^{d \times d}$ and row-orthonormal $T = [\tau_1, \ldots, \tau_N] \in \mathcal{R}^{d \times N}$, where $T_i = [\tau_{i_1}, \ldots, \tau_{i_{k_i}}]$ with the index set $\{i_1, \ldots, i_{k_i}\}$ determined by the neighborhood of each $x_i$, and $e$ is a vector of all ones.

Two strategies are commonly used for selecting the local neighborhood size $k_i$: one is $k$ nearest neighborhood ($k$-NN with a constant $k$ for all the sample points) and the other is $\epsilon$-neighborhood [3, 6]. The effectiveness of the manifold learning algorithms including LTSA depends on the manner of how the nearby neighborhoods overlap with each other and the variation of the curvature of the manifold and its interplay with the sampling density [4]. We illustrate those issues with two simple examples.

**Example 1.** We sample data points from a half unit circle $x_i = [\cos(t_i), \sin(t_i)]^T$, $i = 1 \ldots, N$. It is easy to see that $t_i$ represent the *arc-length* of the circle. We choose $t_i \in [0, \pi]$ according to

$$t_{i+1} - t_i = 0.1(0.001 + |\cos(t_i)|)$$

starting at $t_1 = 0$, and set $N = 152$ so that $t_N \leq \pi$ and $t_{N+1} > \pi$. Clearly, the half circle has unit curvature everywhere. This is an example of highly-varying sampling density.

**Example 2.** The date set is generated as $x_i = [t_i, 10e^{-t_i^2}]^T$, $i = 1 \ldots, N$, where $t_i \in [-6, 6]$ are uniformly distributed. The curvature of the 1-D curve at parameter value $t$ is given by

$$c_g(t) = \frac{20|1 - 2t^2|e^{-t^2}}{(1 + 40t^2e^{-2t^2})^{3/2}}$$

which changes from $\min_t c_g(t) = 0$ to $\max_t c_g(t) = 20$ over $t \in [-6, 6]$. We set $N = 180$. This is an example of highly-varying curvature.

For the above two data sets, LTSA with constant $k$-NN strategy fails for any reasonable $k$ we have tested. So does LTSA with constant $\epsilon$-neighborhoods. In the first column of Figure 1, we plot these two data sets. The computed coordinates by LTSA with constant $k$-neighborhoods are plotted against the centered arc-length coordinates for a selected range of $k$ (ideally, the plots should display points on a straight line of slops $\pm\pi/4$).

## 2   Adaptive Neighborhood Selection

In this section, we propose a neighborhood contraction and expansion algorithm for adaptively selecting $k_i$ at each sample point $x_i$. We assume that the data are generated from a *parameterized* manifold, $x_i = f(\tau_i)$, $i = 1, \ldots, N$, where $f : \Omega \subset \mathcal{R}^d \rightarrow \mathcal{R}^m$. If $f$ is smooth enough, using first-order Taylor expansion at a *fixed* $\tau$, for a neighboring $\bar{\tau}$, we have

$$f(\bar{\tau}) = f(\tau) + J_f(\tau) \cdot (\bar{\tau} - \tau) + \epsilon(\tau, \bar{\tau}), \tag{2.2}$$

where $J_f(\tau) \in \mathcal{R}^{m \times d}$ is the Jacobi matrix of $f$ at $\tau$ and $\epsilon(\tau, \bar{\tau})$ represents the error term determined by the Hessian of $f$, $\|\epsilon(\tau, \bar{\tau})\| \approx c_f(\tau)\|\bar{\tau} - \tau\|_2^2$, where $c_f(\tau) \geq 0$ represents the curvature of the manifold at $\tau$. Setting $\tau = \tau_i$ and $\bar{\tau} = \tau_{i_j}$ gives

$$x_{i_j} = x_i + J_f(\tau_i) \cdot (\tau_{i_j} - \tau_i) + \epsilon(\tau_i, \tau_{i_j}). \tag{2.3}$$

A point $x_{i_j}$ can be regarded as a neighbor of $x_i$ with respect to the tangent space spanned by the columns of $J_f(\tau_i)$ if

$$\|\tau_{i_j} - \tau_i\|_2 \text{ is small and } \|\epsilon(\tau_i, \tau_{i_j})\|_2 \ll \|J_f(\tau_i) \cdot (\tau_{i_j} - \tau_i)\|_2.$$

The above conditions, however, are difficult to verify in practice since we do not know $J_f(\tau_i)$. To get around this problem, consider an orthogonal basis matrix $Q_i$ of the tangent space spanned by the columns of $J_f(\tau_i)$ which can be approximately computed by the SVD of $X_i - \bar{x}_i e^T$, where $\bar{x}_i$ is the mean of the neighbors $x_{i_j} = f(\tau_{i_j})$, $j = 1, \ldots, k_i$. Note that

$$\bar{x}_i = \frac{1}{k_i} \sum_{j=1}^{k_i} x_{i_j} = x_i + J_f(\tau_i) \cdot (\bar{\tau}_i - \tau_i) + \bar{\epsilon}_i,$$

where $\bar{\epsilon}_i$ is the mean of $\epsilon(\tau_i, \tau_{i_1})$, $\ldots$, $\epsilon(\tau_i, \tau_{i_{k_1}})$. Eliminating $x_i$ in (2.3) by the representation above yields $x_{i_j} = \bar{x}_i + J_f(\tau_i) \cdot (\tau_{i_j} - \bar{\tau}_i) + \epsilon_j^{(i)}$ with $\epsilon_j^{(i)} = \epsilon(\tau_i, \tau_{i_j}) - \bar{\epsilon}_i$. Let $\theta_j^{(i)} = Q_i^T(x_{i_j} - \bar{x}_i)$, we have $x_{i_j} = \bar{x}_i + Q_i \theta_j^{(i)} + \epsilon_j^{(i)}$. Thus, $x_{i_j}$ can be selected as a neighbor of $x_i$ if the orthogonal projection $\theta_j^{(i)}$ is small and

$$\|\epsilon_j^{(i)}\|_2 = \|x_{i_j} - \bar{x}_i - Q_i \theta_j^{(i)}\|_2 \ll \|Q_i \theta_j^{(i)}\|_2 = \|\theta_j^{(i)}\|_2. \tag{2.4}$$

Assume all the $x_{i_j}$ satisfy the above inequality, then we should approximately have

$$\|(I - Q_i Q_i^T)(X_i - x_0 e^T)\|_F \leq \eta \|Q_i^T(X_i - x_0 e^T)\|_F \tag{2.5}$$

We will use (2.5) as a criterion for adaptive neighbor selection, starting with a $K$-NN at each sample point $x_i$ with a large enough initial $K$ and deleting points one by one until (2.5) holds. This process will terminate when the neighborhood size equals $d + k_0$ for some small $k_0$ and (2.5) is not true. In that case, we may need to reselect a $k$-NN that minimizes the ratio $\frac{\|(I-Q_i Q_i^T)(X_i - \bar{x}_i e^T)\|_F}{\|Q_i^T(X_i - \bar{x}_i e^T)\|_F}$ as the neighborhood set as is detailed below.

NEIGHBORHOOD CONTRACTION.

C0. Determine the initial $K$ and $K$-NN neighborhood $X_i^{(K)} = [x_{i_1}, \ldots, x_{i_K}]$ for $x_i$, ordered in non-decreasing distances to $x_i$,

$$\|x_{i_1} - x_i\| \le \|x_{i_2} - x_i\| \le \ldots \le \|x_{i_K} - x_i\|.$$

Set $k = K$.

C1. Let $\bar{x}_i^{(k)}$ be the column mean of $X_i^{(k)}$. Compute the orthogonal basis matrix $Q_i^{(k)}$, the $d$ largest singular vectors of $X_i^{(k)} - \bar{x}_i^{(k)} e^T$. Set $\Theta_i^{(k)} = (Q_i^{(k)})^T (X_i^{(k)} - \bar{x}_i^{(k)} e^T)$.

C2. If $\|X_i^{(k)} - \bar{x}_i^{(k)} e^T - Q_i^{(k)} \Theta_i^{(k)}\|_F < \eta \|\Theta_i^{(k)}\|_F$, then set $X_i = X_i^{(k)}$, $\Theta_i = \Theta_i^{(k)}$, and terminate.

C3. If $k > d + k_0$, then delete the last column of $X_i^{(k)}$ to obtain $X_i^{(k-1)}$, set $k := k - 1$, and go to step C1, otherwise, go to step C4.

C4. Let $k = \arg\min_{d+k_0 \le j \le K} \frac{\|X_i^{(j)} - \bar{x}_i^{(j)} e^T - Q_i^{(j)} \Theta_i^{(j)}\|_F}{\|\Theta_i^{(j)}\|_F}$, and set $X_i = X_i^{(k)}$, $\Theta_i = \Theta_i^{(k)}$.

Step C4 means that if there is no $k$-NN ($k \ge d + k_0$) satisfying (2.5), then the contracted neighborhood $X_i$ should be one that minimizes $\frac{\|X_i - \bar{x}_i e^T - Q_i \Theta_i\|_F}{\|\Theta_i\|_F}$.

Once the contraction step is done we can still add back some of unselected $x_{i_j}$ to increase the overlap of nearby neighborhoods while still keep (2.5) intact. In fact, we can add $x_{i_j}$ if $\|x_{i_j} - \bar{x}_i - Q_i \theta_j\| \le \eta \|\theta_j\|$ which is demonstrated in the following result (we refer to [8] for the proof).

**Theorem 2.1** *Let* $X_i = [x_{i_1}, \ldots, x_{i_k}]$ *satisfy (2.5). Furthermore, we assume*

$$\|x_{i_j} - x_0 - Q_i \theta_j^{(i)}\| \le \eta \|\theta_j^{(i)}\|, \quad j = k+1, \ldots, k+p, \tag{2.6}$$

*where* $\theta_j^{(i)} = Q_i^T (x_{i_j} - x_0)$. *Denote by* $\tilde{x}_i$ *the column mean of the expanded matrix* $\tilde{X}_i = [X_i, x_{i_{k+1}}, \ldots x_{i_{k+p}}]$. *Then for the left-singular vector matrix* $\tilde{Q}_i$ *corresponding to the* $d$ *largest singular values of* $\tilde{X}_i - \tilde{x}_i e^T$,

$$\|(I - \tilde{Q}_i \tilde{Q}_i^T)(\tilde{X}_i - \tilde{x}_i e^T)\|_F \le \eta \big( \|\tilde{Q}_i^T (\tilde{X}_i - \tilde{x}_i e^T)\|_F + \frac{\sqrt{p}}{k+p} \| \sum_{j=k+1}^{k+p} \theta_j^{(i)} \|_2 \big).$$

The above result shows that if the mean of the projections $\theta_j^{(i)}$ of the expanding neighbors is small and/or the number of the expanding points are relatively small, then approximately,

$$\|(I - \tilde{Q}_i \tilde{Q}_i^T)(\tilde{X}_i - \tilde{x}_i e^T)\|_F \le \eta \|\tilde{Q}_i^T (\tilde{X}_i - \tilde{x}_i e^T)\|_F.$$

NEIGHBORHOOD EXPANSION.

E0. Set $k_i$ to be the column number of $X_i$ obtained by the neighborhood contracting step. For $j = k_i + 1, \ldots, K$, compute $\theta_j^{(i)} = Q_i^T (x_{i_j} - \bar{x}_i)$.

E1. Denote by $J_i$ the index subset of $j$'s, $k_i < j \le K$, such that $\|(I - Q_i Q_i^T)(x_{i_j} - \bar{x}_i)\|_2 \le \|\theta_j^{(i)}\|_2$. Expand $X_i$ by adding $x_{i_j}$, $j \in J_i$.

**Example 3.** We construct the data points as $x_i = [\sin(t_i), \cos(t_i), 0.02t_i]^T, i = 1, \ldots, N$, with $t_i \in [0, 4\pi]$ uniformly distributed, which is plotted in the top-left panel in Figure 2.

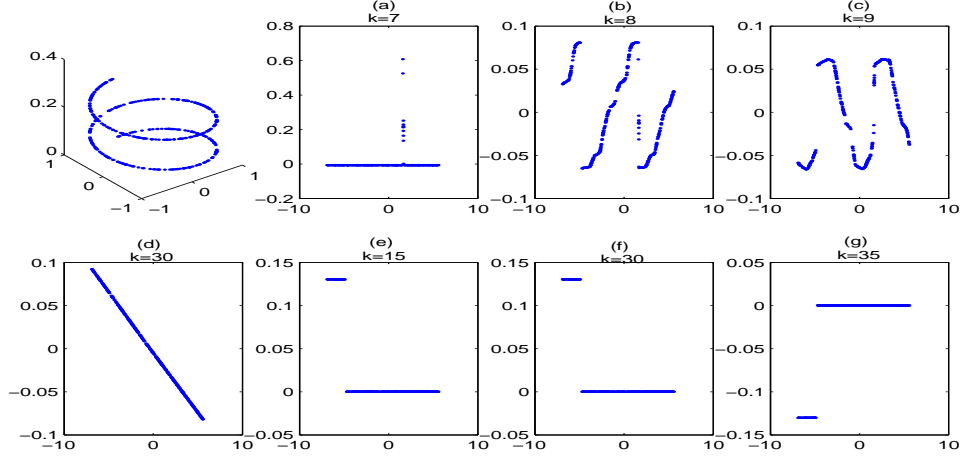

Figure 2: Plots of the data sets (top left), the computed coordinates $\tau_i$ by LTSA vs. the centered arc-length coordinates ($a \sim c$), the computed coordinates $\tau_i$ by LTSA with neighborhood C contraction vs the centered arc-length coordinates ($e \sim g$), and the computed coordinates $\tau_i$ by LTSA with neighborhood contraction and expansion vs. the centered arc-length coordinates (bottom left)

LTSA with constant $k$-NN fails for any $k$: small $k$ leads to lack of necessary overlap among the neighborhoods while for large $k$, the computed tangent space can not represent the local geometry well. In ($a \sim c$) of Figure 2, we plot the coordinates computed by LTSA vs. the arc-length of the curve. Contracting the neighborhoods without expansion also results in bad results, because of small sizes of the resulting neighborhoods, see ($e \sim g$) of Figure 2. Panel (d) of Figure 2 gives an excellent result computed by LTSA with both neighborhood contraction and expansion. We want mention that our adaptive strategies also work well for noisy data sets, we refer the readers to [8] for some examples.

## 3  Alignment incorporating variations of manifold curvature

Let $X_i = [x_{i_1}, \ldots, x_{i_{k_i}}]$ consists of the neighbors determined by the contraction and expansion steps in the above section. In (1.1), we can show that the size of the error term $\|E_i\|_2$ depends on the size of the curvature of manifold at sample point $x_i$ [8]. To make the minimization in (1.1) more uniform, we need to factor out the effect of the variations of the curvature. To this end, we pose the following minimization problem,

$$\min_{T, \{L_i\}} \sum_i \frac{1}{k_i} \|(T_i(I - \frac{1}{k_i}ee^T) - L_i\Theta_i)D_i^{-1}\|_2^2, \qquad (3.7)$$

where $D_i = \text{diag}(\phi(\theta_1^{(i)}), \ldots, \phi(\theta_{k_i}^{(i)}))$, and $\phi(\theta_j^{(i)})$ is proportional to the curvature of the manifold at the parameter value $\theta_i$, the computation of which will be discussed below. For fixed $T$, the optimal $L_i$ is given by $L_i = T_i(I_{k_i} - \frac{1}{k_i}ee^T)\Theta_i^+ = T_i\Theta_i^+$. Substituting it into (3.7), we have the reduced minimization problem

$$\min_T \sum_i \frac{1}{k_i} \|T_i(I_{k_i} - \frac{1}{k_i}ee^T - \Theta_i^+\Theta_i)D_i^{-1}\|_2^2$$

Imposing the normalization condition $TT^T = I$, a solution to the minimization problem above is given by the $d$ eigenvectors corresponding to the second to $(d+1)$st smallest

eigenvalues of the following matrix

$$B \equiv (SW) \operatorname{diag}(D_1^2/k_1, \ldots, D_n^2/k_n)(SW)^T,$$

where $W = (I_{k_i} - \frac{1}{k_i}ee^T)(I_{k_i} - \Theta_i^+\Theta_i)$. Second-order analysis of the error term in (1.1) shows that we can set $\phi_i(\theta_j^{(i)}) = \gamma + c_f(\tau_i)\|\theta_j^{(i)}\|^2$ with a small positive constant $\gamma$ to ensure $\phi_i(\theta_j^{(i)}) > 0$, and $c_f(\tau_i) \geq 0$ represents the mean of curvatures $c_f(\tau_i, \tau_{i_j})$ for all neighbors of $x_i$.

Let $Q_i$ denote the orthonormal matrix of the largest $d$ right singular vectors of $X_i(I - \frac{1}{k_i}ee^T)$. We can approximately compute $c_f(\tau_i)$ as follows.

$$c_f(\tau_i) \approx \frac{1}{k_i - 1} \sum_{\ell=2}^{k_i} \frac{\arccos(\sigma_{\min}(Q_i^T Q_{i_\ell}))}{\|\theta_\ell\|_2}.$$

where $\sigma_{\min}(\cdot)$ is the smallest singular value of a matrix. Then the diagonal weights $\phi(\theta_i)$ can be computed as

$$\phi_i(\theta_j^{(i)}) = \eta + \frac{\|\theta_j\|_2^2}{k_i - 1} \sum_{\ell=2}^{k_i} \frac{\arccos(\sigma_{\min}(Q_i^T Q_{i_\ell}))}{\|\theta_\ell\|_2}.$$

With the above preparation, we are now ready to present the adaptive LTSA algorithm. Given a data set $X = [x_1, \ldots, x_N]$, the approach consists of the following steps:

**Step 1.** Determining the neighborhood $X_i = [x_{i_1}, \ldots, x_{i_{k_i}}]$ for each $x_i$, $i = 1, \ldots, N$, using the neighborhood contraction/expansion steps in Section 2.

**Step 2.** Compute the truncated SVD, say $Q_i\Sigma_i V_i^T$ of $X_i(I - \frac{1}{k_i}ee^T)$ with $d$ columns in both $Q_i$ and $V_i$, the projections $\theta_\ell^{(i)} = Q_i^T(x_{i_\ell} - \bar{x}_i)$ with the mean $\bar{x}_i$ of the neighbors, and denote $\Theta_i = [\theta_1^{(i)}, \ldots, \theta_{k_i}^{(i)}]$.

**Step 3.** Estimate the curvatures as follows. For each $i = 1, \ldots, N$,

$$c_i = \frac{1}{k_i - 1} \sum_{\ell=2}^{k_i-1} \frac{\arccos(\sigma_{\min}(Q_i^T Q_{i_\ell}))}{\|\theta_\ell^{(i)}\|_2},$$

**Step 4.** Construct alignment matrix. For $i = 1, \ldots, N$, set

$$W_i = I_{k_i} - [\frac{1}{\sqrt{k_i}}e, V_i][\frac{1}{\sqrt{k_i}}e, V_i]^T, \quad D_i = \gamma I + \operatorname{diag}(c_i\|\theta_1^{(i)}\|_2^2, \ldots, c_i\|\theta_{k_i}^{(i)}\|_2^2),$$

where $\gamma$ is a small constant number (usually we set $\gamma = 1.0^{-6}$). Set initial $B = 0$. Update $B$ iteratively by

$$B(I_i, I_i) := B(I_i, I_i) + W_i D_i^{-1} D_i^{-1} W_i^T/k_i, \; i = 1, \ldots, N.$$

**Step 5.** Align global coordinates. Compute the $d + 1$ smallest eigen-vectors of $B$ and pick up the eigenvector $[u_2, \ldots, u_{d+1}]$ matrix corresponding to the 2nd to $d + 1$st smallest eigenvalues, and set $T = [u_2, \ldots, u_{d+1}]^T$.

## 4  Experimental Results

In this section, we present several numerical examples to illustrate the performance of the adaptive LTSA algorithm. The test data sets include curves in 2D/3D Euclidean spaces.

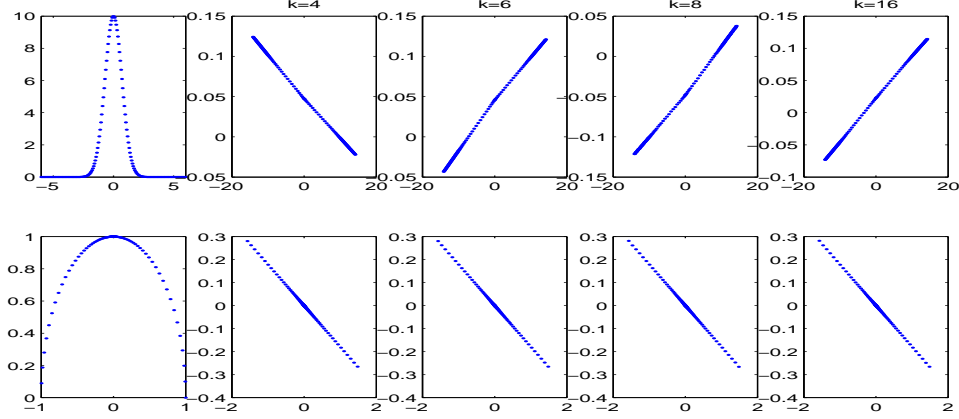

Figure 3: The computed coordinates $\tau_i$ by LTSA taking into account curvature and variable size of neighborhood.

First we apply the adaptive LTSA to the date sets shown in Examples 1 and 2. Adaptive LTSA with different starting $k$'s works every well. See Figure 3. It shows that for these tow data sets, the adaptive LTSA is not sensitive to the choice of the starting $k$ or the variations in sampling densities and manifold curvatures.

Next, we consider the swiss-roll surface defined by $f(s,t) = [s\cos(s), t, s\sin(s)]^T$. It is easy to see that $J_f(s,t)^T J_f(s,t) = \text{diag}(1 + s^2, 1)$. Denoting $s = s(r)$ the inverse transformation of $r = r(s)$ defined by

$$r(s) = \int_0^s \sqrt{1 + \alpha^2}\, d\alpha = \frac{1}{2}(s\sqrt{1 + s^2} + \text{arcsinh}(s)),$$

the swiss-roll surface can be parameterized as

$$\hat{f}(r,t) = [s(t)\cos(s(r)), t, s(r)\sin(s(r))]^T$$

and $\hat{f}$ is isometric with respect to $(r,t)$. In the left figure of Figure 4, we show there is a distortion between the computed coordinates by LTSA with the best-fit neighborhood size (bottom left) and the generating coordinates $(r,t)^T$ (top right). In the right panel of the bottom row of the left figure of Figure 4, we plot the computed coordinates by the adaptive LTSA with initial neighborhood size $k = 30$. (In fact, the adaptive LTSA is insensitive to $k$ and we will get similar results with a larger or smaller initial $k$). We can see that the computed coordinates by the adaptive LTSA can recover the generating coordinates well without much distortion.

Finally we applied both LTSA and the adaptive LTSA to a $2D$ manifold with 3 peaks embedded in a 100 dimensional space. The data points are generated as follows. First we generate $N = 2000$ 3D points, $y_i = (t_i, s_i, h(t_i, s_i))^T$, where $t_i$ and $s_i$ randomly distributed in the interval $[-1.5, 1.5]$ and $h(t,s)$ is defined by

$$h(t,s) = e^{-20t^2 - 20s^2} - e^{-10t^2 - 10(s+1)^2} - e^{-10(1+t)^2 - 10s^2}.$$

Then we embed the 3D points into a 100D space by $x_i^Q = Qy_i$, $x_i^H = Hy_i$, where $Q \in R^{100 \times 3}$ is a random orthonormal matrix resulting in an orthogonal transformation and $H \in R^{100 \times 3}$ a matrix with its singular values uniformly distributed in $(0,1)$ resulting in an affine transformation. In the top row of the right figure of Figure 4, we plot the

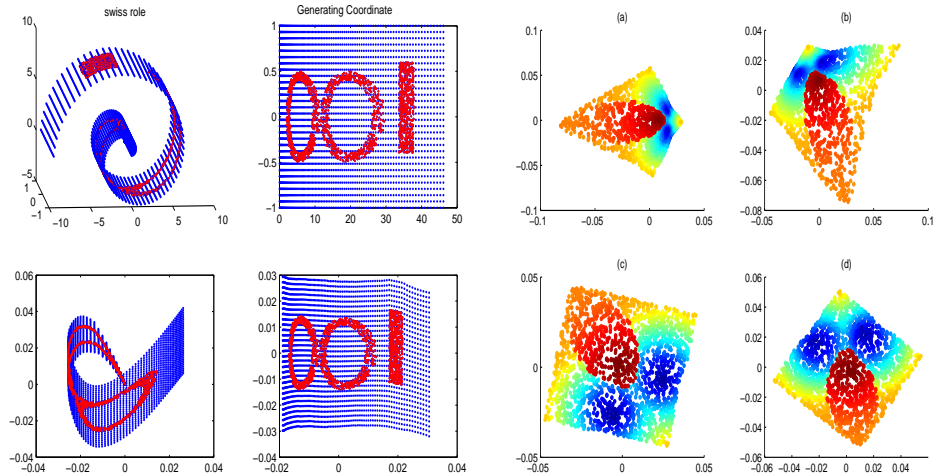

Figure 4: Left figure: 3D swiss-roll and the generating coordinates (top row), computed 2D coordinates by LTSA with the best neighborhood size $k = 15$ (bottom left) and computed 2D coordinates by adaptive LTSA (bottom right). Right figure: coordinates computed by LTSA for the orthogonally embedded 100D data set $\{x_i^Q\}$ (a) and the affinely embedded 100D data set $\{x_i^H\}$ (b), and the coordinates computed by the adaptive LTSA for $\{x_i^Q\}$ (c) and $\{x_i^H\}$ (d).

computed coordinates by LTSA for $x_i^Q$ (shown in (a)) and $x_i^H$ (shown in (b)) with best-fit neighborhood size $k = 15$. We can see the deformations (stretching and compression) are quite prominent. In the bottom row of the right figure of Figure 4, we plot the computed coordinates by the adaptive LTSA for $x_i^Q$ (shown in (c)) and $x_i^H$ (shown in (d)) with initial neighborhood size $k = 15$. It is clear that the adaptive LTSA gives a much better result.

## References

[1] M. Brand. Charting a manifold. *Advances in Neural Information Processing Systems*, 15, MIT Press, 2003.

[2] D. Donoho and C. Grimes. Hessian Eigenmaps: new tools for nonlinear dimensionality reduction. *Proceedings of National Academy of Science*, 5591-5596, 2003.

[3] S. Roweis and L. Saul. Nonlinear dimensionality reduction by locally linear embedding. *Science*, 290: 2323–2326, 2000.

[4] L. Saul and S. Roweis. Think globally, fit locally: unsupervised learning of nonlinear manifolds. *Journal of Machine Learning Research*, 4:119-155, 2003.

[5] E. Teh and S. Roweis. Automatic Alignment of Local Representations. *Advances in Neural Information Processing Systems*, 15, MIT Press, 2003.

[6] J. Tenenbaum, V. De Silva and J. Langford. A global geometric framework for nonlinear dimension reduction. *Science*, 290:2319–2323, 2000.

[7] Z. Zhang and H. Zha. Principal Manifolds and Nonlinear Dimensionality Reduction via Tangent Space Alignment. *SIAM J. Scientific Computing*, 26:313–338, 2004.

[8] J. Wang, Z. Zhang and H. Zha. Adaptive Manifold Learning. Technical Report CSE-04-21, Dept. CSE, Pennsylvania State University, 2004.
